# Statistical Convergence of Kernel CCA

**Kenji Fukumizu**
Institute of Statistical Mathematics
Tokyo 106-8569 Japan
fukumizu@ism.ac.jp

**Francis R. Bach**
Centre de Morphologie Mathematique
Ecole des Mines de Paris, France
francis.bach@mines.org

**Arthur Gretton**
Max Planck Institute for Biological Cybernetics
72076 Tübingen, Germany
arthur.gretton@tuebingen.mpg.de

## Abstract

While kernel canonical correlation analysis (kernel CCA) has been applied in many problems, the asymptotic convergence of the functions estimated from a finite sample to the true functions has not yet been established. This paper gives a rigorous proof of the statistical convergence of kernel CCA and a related method (NOCCO), which provides a theoretical justification for these methods. The result also gives a sufficient condition on the decay of the regularization coefficient in the methods to ensure convergence.

## 1 Introduction

Kernel canonical correlation analysis (kernel CCA) has been proposed as a nonlinear extension of CCA [1, 11, 3]. Given two random variables, kernel CCA aims at extracting the information which is shared by the two random variables, and has been successfully applied in various practical contexts. More precisely, given two random variables $X$ and $Y$, the purpose of kernel CCA is to provide nonlinear mappings $f(X)$ and $g(Y)$ such that their correlation is maximized.

As in many statistical methods, the desired functions are in practice estimated from a finite sample. Thus, the convergence of the estimated functions to the population ones with increasing sample size is very important to justify the method. Since the goal of kernel CCA is to estimate a pair of functions, the convergence should be evaluated in an appropriate functional norm: thus, we need tools from functional analysis to characterize the type of convergence.

The purpose of this paper is to rigorously prove the statistical convergence of kernel CCA, and of a related method. The latter uses a NOrmalized Cross-Covariance Operator, and we call it NOCCO for short. Both kernel CCA and NOCCO require a regularization coefficient to enforce smoothness of the functions in the finite sample case (thus avoiding a trivial solution), but the decay of this regularisation with increased sample size has not yet been established. Our main theorems give a sufficient condition on the decay of the regularization coefficient for the finite sample

estimates to converge to the desired functions in the population limit. Another important issue in establishing the convergence is an appropriate distance measure for functions. For NOCCO, we obtain convergence in the norm of reproducing kernel Hilbert spaces (RKHS) [2]. This norm is very strong: if the positive definite (p.d.) kernels are continuous and bounded, it is stronger than the uniform norm in the space of continuous functions, and thus the estimated functions converge uniformly to the desired ones. For kernel CCA, we show convergence in the $L_2$ norm, which is a standard distance measure for functions. We also discuss the relation between our results and two relevant studies: COCO [9] and CCA on curves [10].

## 2 Kernel CCA and related methods

In this section, we review kernel CCA as presented by [3], and then formulate it with covariance operators on RKHS. In this paper, a Hilbert space always refers to a separable Hilbert space, and an operator to a linear operator. $\|T\|$ denotes the operator norm $\sup_{\|\varphi\|=1} \|T\varphi\|$, and $\mathcal{R}(T)$ denotes the range of an operator $T$.

Throughout this paper, $(\mathcal{H}_\mathcal{X}, k_\mathcal{X})$ and $(\mathcal{H}_\mathcal{Y}, k_\mathcal{Y})$ are RKHS of functions on measurable spaces $\mathcal{X}$ and $\mathcal{Y}$, respectively, with measurable p.d. kernels $k_\mathcal{X}$ and $k_\mathcal{Y}$. We consider a random vector $(X, Y) : \Omega \to \mathcal{X} \times \mathcal{Y}$ with distribution $P_{XY}$. The marginal distributions of $X$ and $Y$ are denoted $P_X$ and $P_Y$. We always assume

$$E_X[k_\mathcal{X}(X, X)] < \infty \quad \text{and} \quad E_Y[k_\mathcal{Y}(Y, Y)] < \infty. \tag{1}$$

Note that under this assumption it is easy to see $\mathcal{H}_\mathcal{X}$ and $\mathcal{H}_\mathcal{Y}$ are continuously included in $L_2(P_X)$ and $L_2(P_Y)$, respectively, where $L_2(\mu)$ denotes the Hilbert space of square integrable functions with respect to the measure $\mu$.

### 2.1 CCA in reproducing kernel Hilbert spaces

Classical CCA provides the linear mappings $a^T X$ and $b^T Y$ that achieve maximum correlation. Kernel CCA extends this by looking for functions $f$ and $g$ such that $f(X)$ and $g(Y)$ have maximal correlation. More precisely, kernel CCA solves

$$\max_{f \in \mathcal{H}_\mathcal{X}, g \in \mathcal{H}_\mathcal{Y}} \frac{\text{Cov}[f(X), g(Y)]}{\text{Var}[f(X)]^{1/2} \text{Var}[g(Y)]^{1/2}}. \tag{2}$$

In practice, we have to estimate the desired function from a finite sample. Given an i.i.d. sample $(X_1, Y_1), \ldots, (X_n, Y_n)$ from $P_{XY}$, an empirical solution of Eq. (2) is

$$\max_{f \in \mathcal{H}_\mathcal{X}, g \in \mathcal{H}_\mathcal{Y}} \frac{\widehat{\text{Cov}}[f(X), g(Y)]}{\left(\widehat{\text{Var}}[f(X)] + \varepsilon_n \|f\|_{\mathcal{H}_\mathcal{X}}^2\right)^{1/2} \left(\widehat{\text{Var}}[g(Y)] + \varepsilon_n \|g\|_{\mathcal{H}_\mathcal{Y}}^2\right)^{1/2}}, \tag{3}$$

where $\widehat{\text{Cov}}$ and $\widehat{\text{Var}}$ denote the empirical covariance and variance, such as

$$\widehat{\text{Cov}}[f(X), g(Y)] = \frac{1}{n}\sum_{i=1}^{n}\left(f(X_i) - \frac{1}{n}\sum_{j=1}^{n}f(X_j)\right)\left(g(Y_i) - \frac{1}{n}\sum_{j=1}^{n}g(Y_j)\right).$$

The positive constant $\varepsilon_n$ is a regularization coefficient. As we shall see, the regularization terms $\varepsilon_n \|f\|_{\mathcal{H}_\mathcal{X}}^2$ and $\varepsilon_n \|g\|_{\mathcal{H}_\mathcal{Y}}^2$ make the problem well-formulated statistically, enforce smoothness, and enable operator inversion, as in Tikhonov regularization.

### 2.2 Representation with cross-covariance operators

Kernel CCA and related methods can be formulated using covariance operators [4, 7, 8], which make theoretical discussions easier. It is known that there exists a unique *cross-covariance operator* $\Sigma_{YX} : \mathcal{H}_\mathcal{X} \to \mathcal{H}_\mathcal{Y}$ for $(X, Y)$ such that

$$\langle g, \Sigma_{YX} f \rangle_{\mathcal{H}_\mathcal{Y}} = E_{XY}\left[(f(X) - E_X[f(X)])(g(Y) - E_Y[g(Y)])\right] \quad (= \text{Cov}[f(X), g(Y)])$$

holds for all $f \in \mathcal{H}_{\mathcal{X}}$ and $g \in \mathcal{H}_{\mathcal{Y}}$. The cross covariance operator represents the covariance of $f(X)$ and $g(Y)$ as a bilinear form of $f$ and $g$. In particular, if $Y$ is equal to $X$, the self-adjoint operator $\Sigma_{XX}$ is called the *covariance operator*.

Let $(X_1, Y_1), \ldots, (X_n, Y_n)$ be i.i.d. random vectors on $\mathcal{X} \times \mathcal{Y}$ with distribution $P_{XY}$. The *empirical cross-covariance operator* $\widehat{\Sigma}_{YX}^{(n)}$ is defined by the cross-covariance operator with the empirical distribution $\frac{1}{n} \sum_{i=1}^{n} \delta_{X_i} \delta_{Y_i}$. By definition, for any $f \in \mathcal{H}_{\mathcal{X}}$ and $g \in \mathcal{H}_{\mathcal{Y}}$, the operator $\widehat{\Sigma}_{YX}^{(n)}$ gives the empirical covariance as follows;

$$\langle g, \widehat{\Sigma}_{YX}^{(n)} f \rangle_{\mathcal{H}_{\mathcal{Y}}} = \widehat{\text{Cov}}[f(X), g(Y)].$$

Let $Q_X$ and $Q_Y$ be the orthogonal projections which respectively map $\mathcal{H}_{\mathcal{X}}$ onto $\overline{\mathcal{R}(\Sigma_{XX})}$ and $\mathcal{H}_{\mathcal{Y}}$ onto $\overline{\mathcal{R}(\Sigma_{YY})}$. It is known [4] that $\Sigma_{YX}$ can be represented as

$$\Sigma_{YX} = \Sigma_{YY}^{1/2} V_{YX} \Sigma_{XX}^{1/2}, \tag{4}$$

where $V_{YX} : \mathcal{H}_{\mathcal{X}} \to \mathcal{H}_{\mathcal{Y}}$ is a unique bounded operator such that $\|V_{YX}\| \leq 1$ and $V_{YX} = Q_Y V_{YX} Q_X$. We often write $V_{YX}$ as $\Sigma_{YY}^{-1/2} \Sigma_{YX} \Sigma_{XX}^{-1/2}$ in an abuse of notation, even when $\Sigma_{XX}^{-1/2}$ or $\Sigma_{YY}^{-1/2}$ are not appropriately defined as operators.

With cross-covariance operators, the kernel CCA problem can be formulated as

$$\sup_{f \in \mathcal{H}_{\mathcal{X}}, g \in \mathcal{H}_{\mathcal{Y}}} \langle g, \Sigma_{YX} f \rangle_{\mathcal{H}_{\mathcal{Y}}} \quad \text{subject to} \quad \begin{cases} \langle f, \Sigma_{XX} f \rangle_{\mathcal{H}_{\mathcal{X}}} = 1, \\ \langle g, \Sigma_{YY} g \rangle_{\mathcal{H}_{\mathcal{Y}}} = 1. \end{cases} \tag{5}$$

As with classical CCA, the solution of Eq. (5) is given by the eigenfunctions corresponding to the largest eigenvalue of the following generalized eigenproblem:

$$\begin{pmatrix} O & \Sigma_{XY} \\ \Sigma_{YX} & O \end{pmatrix} \begin{pmatrix} f \\ g \end{pmatrix} = \rho_1 \begin{pmatrix} \Sigma_{XX} & O \\ O & \Sigma_{YY} \end{pmatrix} \begin{pmatrix} f \\ g \end{pmatrix}. \tag{6}$$

Similarly, the empirical estimator in Eq. (3) is obtained by solving

$$\sup_{f \in \mathcal{H}_{\mathcal{X}}, g \in \mathcal{H}_{\mathcal{Y}}} \langle g, \widehat{\Sigma}_{YX}^{(n)} f \rangle_{\mathcal{H}_{\mathcal{Y}}} \quad \text{subject to} \quad \begin{cases} \langle f, (\widehat{\Sigma}_{XX}^{(n)} + \varepsilon_n I) f \rangle_{\mathcal{H}_{\mathcal{X}}} = 1, \\ \langle g, (\widehat{\Sigma}_{YY}^{(n)} + \varepsilon_n I) g \rangle_{\mathcal{H}_{\mathcal{Y}}} = 1. \end{cases} \tag{7}$$

Let us assume that the operator $V_{YX}$ is compact,[1] and let $\phi$ and $\psi$ be the unit eigenfunctions of $V_{YX}$ corresponding to the largest singular value; that is,

$$\langle \psi, V_{YX} \phi \rangle_{\mathcal{H}_{\mathcal{Y}}} = \max_{f \in \mathcal{H}_{\mathcal{X}}, g \in \mathcal{H}_{\mathcal{Y}}, \|f\|_{\mathcal{H}_{\mathcal{X}}} = \|g\|_{\mathcal{H}_{\mathcal{Y}}} = 1} \langle g, V_{YX} f \rangle_{\mathcal{H}_{\mathcal{Y}}}. \tag{8}$$

Given $\phi \in \mathcal{R}(\Sigma_{XX})$ and $\psi \in \mathcal{R}(\Sigma_{YY})$, the kernel CCA solution in Eq. (6) is

$$f = \Sigma_{XX}^{-1/2} \phi, \qquad g = \Sigma_{YY}^{-1/2} \psi. \tag{9}$$

In the empirical case, let $\widehat{\phi}_n \in \mathcal{H}_{\mathcal{X}}$ and $\widehat{\psi}_n \in \mathcal{H}_{\mathcal{Y}}$ be the unit eigenfunctions corresponding to the largest singular value of the finite rank operator

$$\widehat{V}_{YX}^{(n)} := \left( \widehat{\Sigma}_{YY}^{(n)} + \varepsilon_n I \right)^{-1/2} \widehat{\Sigma}_{YX}^{(n)} \left( \widehat{\Sigma}_{XX}^{(n)} + \varepsilon_n I \right)^{-1/2}. \tag{10}$$

As in Eq. (9), the empirical estimators $\widehat{f}_n$ and $\widehat{g}_n$ in Eq. (7) are equal to

$$\widehat{f}_n = (\widehat{\Sigma}_{XX}^{(n)} + \varepsilon_n I)^{-1/2} \widehat{\phi}_n, \qquad \widehat{g}_n = (\widehat{\Sigma}_{YY}^{(n)} + \varepsilon_n I)^{-1/2} \widehat{\psi}_n. \tag{11}$$

Note that all the above empirical operators and the estimators can be expressed in terms of *Gram matrices*. The solutions $\widehat{f}_n$ and $\widehat{g}_n$ are exactly the same as those given in [3], and are obtained by linear combinations of $k_{\mathcal{X}}(\cdot, X_i) - \frac{1}{n}\sum_{j=1}^{n}k_{\mathcal{X}}(\cdot, X_j)$ and $k_{\mathcal{Y}}(\cdot, Y_i) - \frac{1}{n}\sum_{j=1}^{n}k_{\mathcal{Y}}(\cdot, Y_j)$. The functions $\widehat{\phi}_n$ and $\widehat{\psi}_n$ are obtained similarly.

There exist additional, related methods to extract nonlinear dependence. The constrained covariance (COCO) [9] uses the unit eigenfunctions of $\Sigma_{YX}$;

$$\max_{\substack{f\in\mathcal{H}_{\mathcal{X}}, g\in\mathcal{H}_{\mathcal{Y}}\\ \|f\|_{\mathcal{H}_{\mathcal{X}}}=\|g\|_{\mathcal{H}_{\mathcal{Y}}}=1}} \langle g, \Sigma_{YX}f\rangle_{\mathcal{H}_{\mathcal{Y}}} = \max_{\substack{f\in\mathcal{H}_{\mathcal{X}}, g\in\mathcal{H}_{\mathcal{Y}}\\ \|f\|_{\mathcal{H}_{\mathcal{X}}}=\|g\|_{\mathcal{H}_{\mathcal{Y}}}=1}} \mathrm{Cov}[f(X), g(Y)].$$

The statistical convergence of COCO has been proved in [8]. Instead of normalizing the covariance by the variances, COCO normalizes it by the RKHS norms of $f$ and $g$. Kernel CCA is a more direct nonlinear extension of CCA than COCO. COCO tends to find functions with large variance for $f(X)$ and $g(Y)$, which may not be the most correlated features. On the other hand, kernel CCA may encounter situations where it finds functions with moderately large covariance but very small variance for $f(X)$ or $g(Y)$, since $\Sigma_{XX}$ and $\Sigma_{YY}$ can have arbitrarily small eigenvalues.

A possible compromise is to use $\phi$ and $\psi$ for $V_{YX}$, the NOrmalized Cross-Covariance Operator (NOCCO). While the statistical meaning of NOCCO is not as direct as kernel CCA, it can incorporate the normalization by $\Sigma_{XX}$ and $\Sigma_{YY}$. We will establish the convergence of kernel CCA and NOCCO in Section 3.

## 3 Main theorems: convergence of kernel CCA and NOCCO

We show the convergence of NOCCO in the RKHS norm, and the kernel CCA in $L_2$ sense. The results may easily be extended to the convergence of the eigenspace corresponding to the $m$-th largest eigenvalue.

**Theorem 1.** *Let $(\varepsilon_n)_{n=1}^{\infty}$ be a sequence of positive numbers such that*

$$\lim_{n\to\infty}\varepsilon_n = 0, \qquad \lim_{n\to\infty}n^{1/3}\varepsilon_n = \infty. \tag{12}$$

*Assume $V_{YX}$ is compact, and the eigenspaces given by Eq. (8) are one-dimensional. Let $\phi$, $\psi$, $\widehat{\phi}_n$, and $\widehat{\psi}_n$ be the unit eigenfunctions of Eqs. (8) and (10). Then*

$$|\langle\widehat{\phi}_n, \phi\rangle_{\mathcal{H}_{\mathcal{X}}}| \to 1, \qquad |\langle\widehat{\psi}_n, \psi\rangle_{\mathcal{H}_{\mathcal{Y}}}| \to 1$$

*in probability, as $n$ goes to infinity.*

**Theorem 2.** *Let $(\varepsilon_n)_{n=1}^{\infty}$ be a sequence of positive numbers which satisfies Eq. (12). Assume that $\phi$ and $\psi$ are included in $\mathcal{R}(\Sigma_{XX})$ and $\mathcal{R}(\Sigma_{YY})$, respectively, and that $V_{YX}$ is compact. Then, for $f, g, \widehat{f}_n$, and $\widehat{g}_n$ in Eqs.(9), (11), we have*

$$\left\|(\widehat{f}_n - E_X[\widehat{f}_n(X)]) - (f - E_X[f(X)])\right\|_{L_2(P_X)} \to 0,$$

$$\left\|(\widehat{g}_n - E_Y[\widehat{g}_n(Y)]) - (g - E_Y[g(Y)])\right\|_{L_2(P_Y)} \to 0$$

*in probability, as n goes to infinity.*

The convergence of NOCCO in the RKHS norm is a very strong result. If $k_{\mathcal{X}}$ and $k_{\mathcal{Y}}$ are continuous and bounded, the RKHS norm is stronger than the uniform norm of the continuous functions. In such cases, Theorem 1 implies $\widehat{\phi}_n$ and $\widehat{\psi}_n$ converge uniformly to $\phi$ and $\psi$, respectively. This uniform convergence is useful in practice, because in many applications the function value at each point is important.

For any complete orthonormal systems (CONS) $\{\phi_i\}_{i=1}^{\infty}$ of $\mathcal{H}_{\mathcal{X}}$ and $\{\psi_i\}_{i=1}^{\infty}$ of $\mathcal{H}_{\mathcal{Y}}$, the compactness assumption on $V_{YX}$ requires that the correlation of $\Sigma_{XX}^{-1/2}\phi_i(X)$ and $\Sigma_{YY}^{-1/2}\psi_i(Y)$ decay to zero as $i \to \infty$. This is not necessarily satisfied in general. A trivial example is the case of variables with $Y = X$, in which $V_{YX} = I$ is not compact. In this case, NOCCO is solved by an arbitrary function. Moreover, the kernel CCA does not have solutions, if $\Sigma_{XX}$ has arbitrarily small eigenvalues.

Leurgans et al. ([10]) discuss CCA on curves, which are represented by stochastic processes on an interval, and use the Sobolev space of functions with square integrable second derivative. Since the Sobolev space is a RKHS, their method is an example of kernel CCA. They also show the convergence of estimators under the condition $n^{1/2}\varepsilon_n \to \infty$. Although the proof can be extended to a general RKHS, convergence is measured by the correlation,

$$\frac{|\langle \widehat{f}_n, \Sigma_{XX}f \rangle_{\mathcal{H}_{\mathcal{X}}}|}{(\langle \widehat{f}_n, \Sigma_{XX}\widehat{f}_n \rangle_{\mathcal{H}_{\mathcal{X}}})^{1/2}(\langle f, \Sigma_{XX}f \rangle_{\mathcal{H}_{\mathcal{X}}})^{1/2}} \quad \to \quad 1,$$

which is weaker than the $L_2$ convergence in Theorem 2. In fact, using $\langle f, \Sigma_{XX}f \rangle_{\mathcal{H}_{\mathcal{X}}} = 1$, it is easy to derive the above convergence from Theorem 2. On the other hand, convergence of the correlation does not necessarily imply $\langle (\widehat{f}_n - f), \Sigma_{XX}(\widehat{f}_n - f) \rangle_{\mathcal{H}_{\mathcal{X}}} \to 0$. From the equality

$$\langle (\widehat{f}_n - f), \Sigma_{XX}(\widehat{f}_n - f) \rangle_{\mathcal{H}_{\mathcal{X}}} = (\langle \widehat{f}_n, \Sigma_{XX}\widehat{f}_n \rangle_{\mathcal{H}_{\mathcal{X}}}^{1/2} - \langle f, \Sigma_{XX}f \rangle_{\mathcal{H}_{\mathcal{X}}}^{1/2})^2$$

$$+ 2\{1 - \langle \widehat{f}_n, \Sigma_{XX}f \rangle_{\mathcal{H}_{\mathcal{X}}}/(\|\Sigma_{XX}^{1/2}\widehat{f}_n\|_{\mathcal{H}_{\mathcal{X}}}\|\Sigma_{XX}^{1/2}f\|_{\mathcal{H}_{\mathcal{X}}})\}\|\Sigma_{XX}^{1/2}\widehat{f}_n\|_{\mathcal{H}_{\mathcal{X}}}\|\Sigma_{XX}^{1/2}f\|_{\mathcal{H}_{\mathcal{X}}},$$

we require $\langle \widehat{f}_n, \Sigma_{XX}\widehat{f}_n \rangle_{\mathcal{H}_{\mathcal{X}}} \to \langle f, \Sigma_{XX}f \rangle_{\mathcal{H}_{\mathcal{X}}} = 1$ in order to guarantee the left hand side converges to zero. However, with the normalization $\langle \widehat{f}_n, (\widehat{\Sigma}_{XX}^{(n)} + \varepsilon_n I)\widehat{f}_n \rangle_{\mathcal{H}_{\mathcal{X}}} = 1$, convergence of $\langle \widehat{f}_n, \Sigma_{XX}\widehat{f}_n \rangle_{\mathcal{H}_{\mathcal{X}}}$ is not clear. We use the stronger assumption $n^{1/3}\varepsilon_n \to \infty$ to prove $\langle (\widehat{f}_n - f), \Sigma_{XX}(\widehat{f}_n - f) \rangle_{\mathcal{H}_{\mathcal{X}}} \to 0$ in Theorem 2.

## 4 Outline of the proof of the main theorems

We show only the outline of the proof in this paper. See [6] for the detail.

### 4.1 Preliminary lemmas

We introduce some definitions for our proofs. Let $\mathcal{H}_1$ and $\mathcal{H}_2$ be Hilbert spaces. An operator $T : \mathcal{H}_1 \to \mathcal{H}_2$ is called *Hilbert-Schmidt* if $\sum_{i=1}^{\infty}\|T\varphi_i\|_{\mathcal{H}_2}^2 < \infty$ for a CONS $\{\varphi_i\}_{i=1}^{\infty}$ of $\mathcal{H}_1$. Obviously $\|T\| \leq \|T\|_{HS}$. For Hilbert-Schmidt operators, the Hilbert-Schmidt norm and inner product are defined as

$$\|T\|_{HS}^2 = \sum_{i=1}^{\infty}\|T\varphi_i\|_{\mathcal{H}_2}^2, \qquad \langle T_1, T_2 \rangle_{HS} = \sum_{i=1}^{\infty}\langle T_1\varphi_i, T_2\varphi_i \rangle_{\mathcal{H}_2}.$$

These definitions are independent of the CONS. For more details, see [5] and [8].

For a Hilbert space $\mathcal{F}$, a Borel measurable map $F : \Omega \to \mathcal{F}$ from a measurable space $F$ is called a *random element* in $\mathcal{F}$. For a random element $F$ in $\mathcal{F}$ with $E\|F\| < \infty$, there exists a unique element $E[F] \in \mathcal{F}$, called the *expectation* of $F$, such that

$$\langle E[F], g \rangle_{\mathcal{H}} = E[\langle F, g \rangle_{\mathcal{F}}] \qquad (\forall g \in \mathcal{F})$$

holds. If random elements $F$ and $G$ in $\mathcal{F}$ satisfy $E[\|F\|^2] < \infty$ and $E[\|G\|^2] < \infty$, then $\langle F, G \rangle_{\mathcal{F}}$ is integrable. Moreover, if $F$ and $G$ are independent, we have

$$E[\langle F, G \rangle_{\mathcal{F}}] = \langle E[F], E[G] \rangle_{\mathcal{F}}. \tag{13}$$

It is easy to see under the condition Eq. (1), the random element $k_\mathcal{X}(\cdot, X)k_\mathcal{Y}(\cdot, Y)$ in the direct product $\mathcal{H}_\mathcal{X} \otimes \mathcal{H}_\mathcal{Y}$ is integrable, i.e. $E[\|k_\mathcal{X}(\cdot, X)k_\mathcal{Y}(\cdot, Y)\|_{\mathcal{H}_\mathcal{X} \otimes \mathcal{H}_\mathcal{Y}}] < \infty$. Combining Lemma 1 in [8] and Eq. (13), we obtain the following lemma.

**Lemma 3.** *The cross-covariance operator $\Sigma_{YX}$ is Hilbert-Schmidt, and*

$$\|\Sigma_{YX}\|_{HS}^2 = \left\| E_{YX}\left[ \left( k_\mathcal{X}(\cdot, X) - E_X[k_\mathcal{X}(\cdot, X)] \right) \left( k_\mathcal{Y}(\cdot, Y) - E_Y[k_\mathcal{Y}(\cdot, Y)] \right) \right] \right\|_{\mathcal{H}_\mathcal{X} \otimes \mathcal{H}_\mathcal{Y}}^2.$$

The law of large numbers implies $\lim_{n \to \infty} \langle g, \widehat{\Sigma}_{YX}^{(n)} f \rangle_{\mathcal{H}_\mathcal{Y}} = \langle g, \Sigma_{YX} f \rangle_{\mathcal{H}_\mathcal{Y}}$ for each $f$ and $g$ in probability. The following lemma shows a much stronger uniform result.

**Lemma 4.**
$$\left\| \widehat{\Sigma}_{YX}^{(n)} - \Sigma_{YX} \right\|_{HS} = O_p(n^{-1/2}) \quad (n \to \infty).$$

*Proof.* Write for simplicity $F = k_\mathcal{X}(\cdot, X) - E_X[k_\mathcal{X}(\cdot, X)]$, $G = k_\mathcal{Y}(\cdot, Y) - E_Y[k_\mathcal{Y}(\cdot, Y)]$, $F_i = k_\mathcal{X}(\cdot, X_i) - E_X[k_\mathcal{X}(\cdot, X)]$, and $G_i = k_\mathcal{Y}(\cdot, Y_i) - E_Y[k_\mathcal{Y}(\cdot, Y)]$. Then, $F, F_1, \ldots, F_n$ are i.i.d. random elements in $\mathcal{H}_\mathcal{X}$, and a similar property also holds for $G, G_1, \ldots, G_n$. Lemma 3 and the same argument as its proof implies

$$\left\| \widehat{\Sigma}_{YX}^{(n)} \right\|_{HS}^2 = \left\| \tfrac{1}{n} \sum_{i=1}^n \left( F_i - \tfrac{1}{n} \sum_{j=1}^n F_j \right) \left( G_i - \tfrac{1}{n} \sum_{j=1}^n G_j \right) \right\|_{\mathcal{H}_\mathcal{X} \otimes \mathcal{H}_\mathcal{Y}}^2,$$

$$\langle \Sigma_{YX}, \widehat{\Sigma}_{YX}^{(n)} \rangle_{HS} = \left\langle E[FG], \tfrac{1}{n} \sum_{i=1}^n \left( F_i - \tfrac{1}{n} \sum_{j=1}^n F_j \right) \left( G_i - \tfrac{1}{n} \sum_{j=1}^n G_j \right) \right\rangle_{\mathcal{H}_\mathcal{X} \otimes \mathcal{H}_\mathcal{Y}}.$$

From these equations, we have

$$\left\| \widehat{\Sigma}_{YX}^{(n)} - \Sigma_{YX} \right\|_{HS}^2 = \left\| \tfrac{1}{n} \sum_{i=1}^n \left( F_i - \tfrac{1}{n} \sum_{j=1}^n F_j \right) \left( G_i - \tfrac{1}{n} \sum_{j=1}^n G_j \right) - E[FG] \right\|_{\mathcal{H}_\mathcal{X} \otimes \mathcal{H}_\mathcal{Y}}^2$$

$$= \left\| \tfrac{1}{n}\left(1 - \tfrac{1}{n}\right) \sum_{i=1}^n F_i G_i - \tfrac{1}{n^2} \sum_{i=1}^n \sum_{j \neq i} (F_i G_j + F_j G_i) - E[FG] \right\|_{\mathcal{H}_\mathcal{X} \otimes \mathcal{H}_\mathcal{Y}}^2.$$

Using $E[F_i] = E[G_i] = 0$ and $E[F_i G_j F_k G_\ell] = 0$ for $i \neq j, \{k, \ell\} \neq \{i, j\}$, we have

$$E\left\| \widehat{\Sigma}_{YX}^{(n)} - \Sigma_{YX} \right\|_{HS}^2 = \tfrac{1}{n} E\left[ \|FG\|_{\mathcal{H}_\mathcal{X} \otimes \mathcal{H}_\mathcal{Y}}^2 \right] - \tfrac{1}{n} \|E[FG]\|_{\mathcal{H}_\mathcal{X} \otimes \mathcal{H}_\mathcal{Y}}^2 + O(1/n^2).$$

The proof is completed by Chebyshev's inequality. $\qquad \square$

The following two lemmas are essential parts of the proof of the main theorems.

**Lemma 5.** *Let $\varepsilon_n$ be a positive number such that $\varepsilon_n \to 0$ $(n \to \infty)$. Then*

$$\left\| \widehat{V}_{YX}^{(n)} - (\Sigma_{YY} + \varepsilon_n I)^{-1/2} \Sigma_{YX} (\Sigma_{XX} + \varepsilon_n I)^{-1/2} \right\| = O_p(\varepsilon_n^{-3/2} n^{-1/2}).$$

*Proof.* The operator on the left hand side is equal to

$$\left\{ (\widehat{\Sigma}_{YY}^{(n)} + \varepsilon_n I)^{-1/2} - (\Sigma_{YY} + \varepsilon_n I)^{-1/2} \right\} \widehat{\Sigma}_{YX}^{(n)} (\widehat{\Sigma}_{XX}^{(n)} + \varepsilon_n I)^{-1/2}$$

$$+ (\Sigma_{YY} + \varepsilon_n I)^{-1/2} \left\{ \widehat{\Sigma}_{YX}^{(n)} - \Sigma_{YX} \right\} (\widehat{\Sigma}_{XX}^{(n)} + \varepsilon_n I)^{-1/2}$$

$$+ (\Sigma_{YY} + \varepsilon_n I)^{-1/2} \Sigma_{YX} \left\{ (\widehat{\Sigma}_{XX}^{(n)} + \varepsilon_n I)^{-1/2} - (\Sigma_{XX} + \varepsilon_n I)^{-1/2} \right\}. \qquad (14)$$

From the equality $A^{-1/2} - B^{-1/2} = A^{-1/2}(B^{3/2} - A^{3/2}) B^{-3/2} + (A - B) B^{-3/2}$, the first term in Eq. (14) is equal to

$$\left\{ (\widehat{\Sigma}_{YY}^{(n)} + \varepsilon_n I)^{-\frac{1}{2}} \left( \Sigma_{YY}^{\frac{3}{2}} - \widehat{\Sigma}_{YY}^{(n)\frac{3}{2}} \right) + \left( \widehat{\Sigma}_{YY}^{(n)} - \Sigma_{YY} \right) \right\} \left( \widehat{\Sigma}_{YY}^{(n)} + \varepsilon_n I \right)^{-\frac{3}{2}} \widehat{\Sigma}_{YX}^{(n)} (\widehat{\Sigma}_{XX}^{(n)} + \varepsilon_n I)^{-\frac{1}{2}}.$$

From $\|(\widehat{\Sigma}_{YY}^{(n)} + \varepsilon_n I)^{-1/2}\| \leq 1/\sqrt{\varepsilon_n}$, $\|(\widehat{\Sigma}_{YY}^{(n)} + \varepsilon_n I)^{-1/2} \widehat{\Sigma}_{YX}^{(n)} (\widehat{\Sigma}_{XX}^{(n)} + \varepsilon_n I)^{-1/2}\| \leq 1$ and Lemma 7, the norm of the above operator is upper-bounded by

$$\tfrac{1}{\varepsilon_n} \left\{ \tfrac{3}{\sqrt{\varepsilon_n}} \max\left\{ \|\Sigma_{YY}\|^{3/2}, \|\widehat{\Sigma}_{YY}^{(n)}\|^{3/2} \right\} + 1 \right\} \|\widehat{\Sigma}_{YY}^{(n)} - \Sigma_{YY}\|.$$

A similar bound applies to the third term of Eq. (14), and the second term is upper-bounded by $\tfrac{1}{\varepsilon_n} \|\Sigma_{YX} - \widehat{\Sigma}_{YX}^{(n)}\|$. Thus, Lemma 4 completes the proof. $\qquad \square$

**Lemma 6.** *Assume $V_{YX}$ is compact. Then, for a sequence $\varepsilon_n \to 0$,*

$$\left\| (\Sigma_{YY} + \varepsilon_n I)^{-1/2} \Sigma_{YX} (\Sigma_{XX} + \varepsilon_n I)^{-1/2} - V_{YX} \right\| \to 0 \quad (n \to \infty).$$

*Proof.* It suffices to prove $\|\{(\Sigma_{YY} + \varepsilon_n I)^{-1/2} - \Sigma_{YY}^{-1/2}\}\Sigma_{YX}(\Sigma_{XX} + \varepsilon_n I)^{-1/2}\|$ and $\|\Sigma_{YY}^{-1/2}\Sigma_{YX}\{(\Sigma_{XX} + \varepsilon_n I)^{-1/2} - \Sigma_{XX}^{-1/2}\}\|$ converge to zero. The former is equal to

$$\left\| \{(\Sigma_{YY} + \varepsilon_n I)^{-1/2}\Sigma_{YY}^{1/2} - I\}V_{YX} \right\|. \tag{15}$$

Note that $\mathcal{R}(V_{YX}) \subset \overline{\mathcal{R}(\Sigma_{YY})}$, as remarked in Section 2.2. Let $v = \Sigma_{YY}u$ be an arbitrary element in $\mathcal{R}(V_{YX}) \cap \mathcal{R}(\Sigma_{YY})$. We have $\|\{(\Sigma_{YY} + \varepsilon_n I)^{-1/2}\Sigma_{YY}^{1/2} - I\}v\|_{\mathcal{H}_\mathcal{Y}} = \|(\Sigma_{YY} + \varepsilon_n I)^{-1/2}\Sigma_{YY}^{1/2}\{\Sigma_{YY}^{1/2} - (\Sigma_{YY} + \varepsilon_n I)^{1/2}\}\Sigma_{YY}^{1/2}u\|_{\mathcal{H}_\mathcal{Y}} \leq \|\Sigma_{YY}^{1/2} - (\Sigma_{YY} + \varepsilon_n I)^{1/2}\| \, \|\Sigma_{YY}^{1/2}u\|_{\mathcal{H}_\mathcal{Y}}$. Since $(\Sigma_{YY} + \varepsilon_n I)^{1/2} \to \Sigma_{YY}^{1/2}$ in norm, we obtain

$$\{(\Sigma_{YY} + \varepsilon_n I)^{-1/2}\Sigma_{YY}^{1/2} - I\}v \to 0 \qquad (n \to \infty) \tag{16}$$

for all $v \in \mathcal{R}(V_{YX}) \cap \mathcal{R}(\Sigma_{YY})$. Because $V_{YX}$ is compact, Lemma 8 in the Appendix shows Eq. (15) converges to zero. The convergence of the second norm is similar. $\square$

## 4.2 Proof of the main theorems

*Proof of Thm. 1.* This follows from Lemmas 5, 6, and Lemma 9 in Appendix. $\square$

*Proof Thm. 2.* We show only the convergence of $\widehat{f}_n$. W.l.o.g, we can assume $\widehat{\phi}_n \to \phi$ in $\mathcal{H}_\mathcal{X}$. From $\|\Sigma_{XX}^{1/2}(\widehat{f}_n - f)\|_{\mathcal{H}_\mathcal{X}}^2 = \|\Sigma_{XX}^{1/2}\widehat{f}_n\|_{\mathcal{H}_\mathcal{X}}^2 - 2\langle \phi, \Sigma_{XX}^{1/2}\widehat{f}_n\rangle_{\mathcal{H}_\mathcal{X}} + \|\phi\|_{\mathcal{H}_\mathcal{X}}^2$, it suffices to show $\Sigma_{XX}^{1/2}\widehat{f}_n$ converges to $\phi$ in probability. We have

$$\|\Sigma_{XX}^{1/2}\widehat{f}_n - \phi\|_{\mathcal{H}_\mathcal{X}} \leq \|\Sigma_{XX}^{1/2}\{(\widehat{\Sigma}_{XX}^{(n)} + \varepsilon_n I)^{-1/2} - (\Sigma_{XX} + \varepsilon_n I)^{-1/2}\}\widehat{\phi}_n\|_{\mathcal{H}_\mathcal{X}}$$
$$+ \|\Sigma_{XX}^{1/2}(\Sigma_{XX} + \varepsilon_n I)^{-1/2}(\widehat{\phi}_n - \phi)\|_{\mathcal{H}_\mathcal{X}} + \|\Sigma_{XX}^{1/2}(\Sigma_{XX} + \varepsilon_n I)^{-1/2}\phi - \phi\|_{\mathcal{H}_\mathcal{X}}.$$

Using the same argument as the bound on the first term in Eq. (14), the first term on the R.H.S of the above inequality is shown to converge to zero. The convergence of the second term is obvious. Using the assumption $\phi \in \mathcal{R}(\Sigma_{XX})$, the same argument as the proof of Eq. (16) applies to the third term, which completes the proof. $\square$

## 5 Concluding remarks

We have established the statistical convergence of kernel CCA and NOCCO, showing that the finite sample estimators of the nonlinear mappings converge to the desired population functions. This convergence is proved in the RKHS norm for NOCCO, and in the $L_2$ norm for kernel CCA. These results give a theoretical justification for using the empirical estimates of NOCCO and kernel CCA in practice.

We have also derived a sufficient condition, $n^{1/3}\varepsilon_n \to \infty$, for the decay of the regularization coefficient $\varepsilon_n$, which ensures the convergence described above. As [10] suggests, the order of the sufficient condition seems to depend on the function norm used to determine convergence. An interesting consideration is whether the order $n^{1/3}\varepsilon_n \to \infty$ can be improved for convergence in the $L_2$ or RKHS norm.

Another question that remains to be addressed is when to use kernel CCA, COCO, or NOCCO in practice. The answer probably depends on the statistical properties of the data. It might consequently be helpful to determine the relation between the spectral properties of the data distribution and the solutions of these methods.

**Acknowledgements**

This work is partially supported by KAKENHI 15700241 and Inamori Foundation.

## Footnotes

[1] A bounded operator $T : \mathcal{H}_1 \to \mathcal{H}_2$ is called *compact* if any bounded sequence $\{u_n\} \subset \mathcal{H}_1$ has a subsequence $\{u_{n'}\}$ such that $Tu_{n'}$ converges in $\mathcal{H}_2$. One of the useful properties of a compact operator is that it admits a singular value decomposition (see [5, 6])

## References

[1] S. Akaho. A kernel method for canonical correlation analysis. *Proc. Intern. Meeting on Psychometric Society (IMPS2001)*, 2001.

[2] N. Aronszajn. Theory of reproducing kernels. *Trans. American Mathematical Society*, 69(3):337–404, 1950.

[3] F. R. Bach and M. I. Jordan. Kernel independent component analysis. *J. Machine Learning Research*, 3:1–48, 2002.

[4] C. R. Baker. Joint measures and cross-covariance operators. *Trans. American Mathematical Society*, 186:273–289, 1973.

[5] N. Dunford and J. T. Schwartz. *Linear Operators, Part II.* Interscience, 1963.

[6] K. Fukumizu, F. R. Bach, and A. Gretton. Consistency of kernel canonical correlation. Research Memorandum 942, Institute of Statistical Mathematics, 2005.

[7] K. Fukumizu, F. R. Bach, and M. I. Jordan. Dimensionality reduction for supervised learning with reproducing kernel Hilbert spaces. *J. Machine Learning Research*, 5:73–99, 2004.

[8] A. Gretton, O. Bousquet, A. Smola, and B. Schölkopf. Measuring statistical dependence with Hilbert-Schmidt norms. Tech Report 140, Max-Planck-Institut für biologische Kybernetik, 2005.

[9] A. Gretton, A. Smola, O. Bousquet, R. Herbrich, B. Schölkopf, and N. Logothetis. Behaviour and convergence of the constrained covariance. Tech Report 128, Max-Planck-Institut für biologische Kybernetik, 2004.

[10] S. Leurgans, R. Moyeed, and B. Silverman. Canonical correlation analysis when the data are curves. *J. Royal Statistical Society, Series B*, 55(3):725–740, 1993.

[11] T. Melzer, M. Reiter, and H. Bischof. Nonlinear feature extraction using generalized canonical correlation analysis. *Proc. Intern. Conf. Artificial Neural Networks (ICANN2001)*, 353–360, 2001.

## A  Lemmas used in the proofs

We list the lemmas used in Section 4. See [6] for the proofs.

**Lemma 7.** *Suppose $A$ and $B$ are positive self-adjoint operators on a Hilbert space such that $0 \leq A \leq \lambda I$ and $0 \leq B \leq \lambda I$ hold for a positive constant $\lambda$. Then*

$$\|A^{3/2} - B^{3/2}\| \leq 3\lambda^{3/2}\|A - B\|.$$

**Lemma 8.** *Let $\mathcal{H}_1$ and $\mathcal{H}_2$ be Hilbert spaces, and $\mathcal{H}_0$ be a dense linear subspace of $\mathcal{H}_2$. Suppose $A_n$ and $A$ are bounded operators on $\mathcal{H}_2$, and $B$ is a compact operator from $\mathcal{H}_1$ to $\mathcal{H}_2$ such that $A_n u \to Au$ for all $u \in \mathcal{H}_0$, and $\sup_n \|A_n\| \leq M$ for some $M > 0$. Then $A_n B$ converges to $AB$ in norm.*

**Lemma 9.** *Let $A$ be a compact positive operator on a Hilbert space $\mathcal{H}$, and $A_n$ ($n \in \mathbb{N}$) be bounded positive operators on $\mathcal{H}$ such that $A_n$ converges to $A$ in norm. Assume the eigenspace of $A$ corresponding to the largest eigenvalue is one-dimensional and spanned by a unit eigenvector $\phi$, and the maximum of the spectrum of $A_n$ is attained by a unit eigenvector $\phi_n$. Then we have $|\langle\phi_n, \phi\rangle_\mathcal{H}| \to 1$ as $n \to \infty$.*